# Transfer Learning by Distribution Matching for Targeted Advertising

**Steffen Bickel, Christoph Sawade, and Tobias Scheffer**
University of Potsdam, Germany
{bickel, sawade, scheffer}@cs.uni-potsdam.de

## Abstract

We address the problem of learning classifiers for several related tasks that may differ in their joint distribution of input and output variables. For each task, small – possibly even empty – labeled samples and large unlabeled samples are available. While the unlabeled samples reflect the target distribution, the labeled samples may be biased. This setting is motivated by the problem of predicting sociodemographic features for users of web portals, based on the content which they have accessed. Here, questionnaires offered to a portion of each portal's users produce biased samples. We derive a transfer learning procedure that produces resampling weights which match the pool of all examples to the target distribution of any given task. Transfer learning enables us to make predictions even for new portals with few or no training data and improves the overall prediction accuracy.

## 1   Introduction

We study a problem setting of *transfer learning* in which classifiers for *multiple tasks* have to be learned from *biased samples*. Some of the multiple tasks will likely relate to one another, but one cannot assume that the tasks share a joint conditional distribution of the class label given the input variables. The challenge of multi-task learning is to come to a good generalization across tasks: each task should benefit from the wealth of data available for the entirety of tasks, but the optimization criterion needs to remain tied to the individual task at hand.

A common method for learning under covariate shift (marginal shift) is to weight the biased training examples by the test-to-training ratio $\frac{p_{\text{test}}(\mathbf{x})}{p_{\text{train}}(\mathbf{x})}$ to match the marginal distribution of the test data [1]. Instead of separately estimating the two potentially high-dimensional densities one can directly estimate the density ratio – by kernel mean matching [2], minimization of the KL-divergence between test and weighted training data [3], or by discrimination of training against test data with a probabilistic classifier [4].

Hierarchical Bayesian models are a standard statistical approach to multi-task learning [5, 6, 7]. Here, a common prior on model parameters across tasks captures the task dependencies. Similar to the idea of learning under marginal shift by weighting the training examples, [8] devise a method for learning under joint shift of covariates and labels over multiple tasks that is based on instance-specific rescaling factors. We generalize this idea to a setting where not only the joint distributions between tasks may differ but also the training and test distribution within each task.

Our work is motivated by the targeted advertising problem for which the goal is to predict sociodemographic features (such as gender, age, or marital status) of web users, based on their surfing history. Many types of products are specifically targeted at clearly defined market segments, and marketing organizations seek to disseminate their message under minimal costs per delivery to a targeted individual. When sociodemographic attributes can be identified, delivering advertisements to users outside the target segment can be avoided. For some campaigns, clicks and resulting on-

line purchases constitute an ultimate success criterion. However, for many campaigns – including campaigns for products that are not typically purchased on the web – the sole goal is to deliver the advertisement to customers in the target segment.

The paper is structured as follows. Section 2 defines the problem setting. In Section 3, we devise our transfer learning model. We empirically study transfer learning for targeted advertising in Section 4 and Section 5 concludes.

## 2   Problem Setting

We consider the following multi-task learning scenario. Each of several tasks $z$ is characterized by an unknown joint distribution $p_{\text{test}}(\mathbf{x}, y|z) = p_{\text{test}}(\mathbf{x}|z)p(y|\mathbf{x}, z)$ over features $\mathbf{x}$ and labels $y$ given the task $z$. The joint distributions of different tasks may differ arbitrarily but usually some tasks have similar distributions. An unlabeled test sample $T = \langle(\mathbf{x}_1, z_1), \ldots, (\mathbf{x}_m, z_m)\rangle$ with examples from different tasks is available. For each test example, attributes $\mathbf{x}_i$ and the originating task $z_i$ are known. The test data for task $z$ are governed by $p_{\text{test}}(\mathbf{x}|z)$.

A labeled training set $L = \langle(\mathbf{x}_{m+1}, y_{m+1}, z_{m+1}), \ldots, (\mathbf{x}_{m+n}, y_{m+n}, z_{m+n})\rangle$ collects examples from several tasks. In addition to $\mathbf{x}_i$ and $z_i$, the label $y_i$ is known for each example. The training data for task $z$ is drawn from a joint distribution $p_{\text{train}}(\mathbf{x}, y|z) = p_{\text{train}}(\mathbf{x}|z)p(y|\mathbf{x}, z)$ that may differ from the test distribution in terms of the marginal distribution $p_{\text{train}}(\mathbf{x}|z)$. The training and test marginals may differ arbitrarily, as long as each $\mathbf{x}$ with positive $p_{\text{test}}(\mathbf{x}|z)$ also has a positive $p_{\text{train}}(\mathbf{x}|z)$. This guarantees that the training distribution covers the entire support of the test distribution for each task. The conditional distribution $p(y|\mathbf{x}, z)$ of test and training data is identical for a given task $z$, but conditionals can differ arbitrarily between tasks. The entire training set over all tasks is governed by the mixed density $p_{\text{train}}(z)p_{\text{train}}(\mathbf{x}, y|z)$. The prior $p_{\text{train}}(z)$ specifies the task proportions. There may be tasks with only a few or no labeled data.

The goal is to learn a hypothesis $f_z : \mathbf{x} \mapsto y$ for each task $z$. This hypothesis $f_z(\mathbf{x})$ should correctly predict the true label $y$ of unseen examples drawn from $p(\mathbf{x}|z)$ for all $z$. That is, it should minimize the expected loss

$$\mathbf{E}_{(\mathbf{x}, y) \sim p_{\text{test}}(\mathbf{x}, y|z)}[\ell(f_z(\mathbf{x}), y)]$$

with respect to the unknown distribution $p_{\text{test}}(\mathbf{x}, y|z)$ for each individual $z$.

This abstract problem setting models the targeted advertising application as follows. The feature vector $\mathbf{x}$ encodes the web surfing behavior of a user of web portal $z$ (task). For a small number of users the sociodemographic target label $y$ (e.g., gender of user) is collected through web surveys. For new portals the number of such labeled training instances is initially small. The sociodemographic labels for all users of all portals are to be predicted. The joint distribution $p_{\text{test}}(\mathbf{x}, y|z)$ can be different between portals since they attract specific populations of users. The training distribution differs from the test distribution because the response to the web surveys is not uniform with respect to the test distribution. Since the completion of surveys cannot be enforced, it is intrinsically impossible to obtain labeled samples that are governed by the test distribution. Therefore, a possible difference between the conditionals $p_{\text{test}}(y|\mathbf{x}, z)$ and $p_{\text{train}}(y|\mathbf{x}, z)$ cannot be reflected in the model.

One reference strategy is to learn individual models for each target task $z$ by minimizing an appropriate loss function on the portion of $L_z = \{(\mathbf{x}_i, y_i, z_i) \in L : z_i = z\}$. This procedure does not exploit data of related tasks. In addition, it minimizes the loss with respect to $p_{\text{train}}(\mathbf{x}, y|z)$; the minimum of this optimization problem will not generally coincide with the minimal loss on $p_{\text{test}}(\mathbf{x}, y|z)$. The other extreme is a one-size-fits-all model $f_*(\mathbf{x})$ trained on the pooled training sample $L$. The training sample may deviate arbitrarily from the target distribution $p_{\text{test}}(\mathbf{x}, y|z)$.

In order to describe the following model accurately, we introduce selector variable $s$ which distinguishes training ($s = 1$) from test distributions ($s = -1$). Symbol $p_{\text{train}}(\mathbf{x}, y|z)$ is a shorthand for $p(\mathbf{x}, y|z, s = 1)$; likewise, $p_{\text{test}}(\mathbf{x}, y|z) = p(\mathbf{x}, y|z, s = -1)$.

## 3   Transfer Learning by Distribution Matching

In learning a classifier $f_t(\mathbf{x})$ for target task $t$, we seek to minimize the loss function with respect to $p_{\text{test}}(\mathbf{x}, y|t) = p(\mathbf{x}, y|t, s = -1)$. Both, $t$ and $z$ are values of the random variable *task*; value $t$

identifies the current target task. Simply pooling the available data for all tasks would create a sample governed by $\sum_z p(z|s=1)p(\mathbf{x},y|z,s=1)$. Our approach is to create a task-specific resampling weight $r_t(\mathbf{x},y)$ for each element of the pool of examples. The resampling weights match the pool distribution to the target distribution $p(\mathbf{x},y|t,s=-1)$. The resampled pool is governed by the correct target distribution, but is larger than the labeled sample of the target task. Instead of sampling from the pool, one can weight the loss incurred by each instance by the resampling weight.

The expected weighted loss with respect to the mixture distribution that governs the pool equals the loss with respect to the target distribution $p(\mathbf{x},y|t,s=-1)$. Equation 1 defines the condition that the resampling weights have to satisfy.

$$\mathbf{E}_{(\mathbf{x},y)\sim p(\mathbf{x},y|t,s=-1)}[\ell(f(\mathbf{x},t),y)] \tag{1}$$
$$= \mathbf{E}_{(\mathbf{x},y)\sim\sum_z p(z|s=1)p(\mathbf{x},y|z,s=1)}[r_t(\mathbf{x},y)\ell(f(\mathbf{x},t),y)]$$

In the following, we will show that

$$r_t(\mathbf{x},y) = \frac{p(\mathbf{x},y|t,s=1)}{\sum_z p(z|s=1)p(\mathbf{x},y|z,s=1)}\frac{p(\mathbf{x}|t,s=-1)}{p(\mathbf{x}|t,s=1)} \tag{2}$$

satisfies Equation 1. Equation 3 expands the expectation and introduces two fractions that equal one. We can factorize $p(\mathbf{x},y|t,s=-1)$ and expand the sum over $z$ in the numerator to run over the entire expression because the integral over $(\mathbf{x},y)$ is independent of $z$ (Equation 4). Equation 5 rearranges some terms and Equation 6 is the expected loss over the distribution of all tasks weighted by $r_t(\mathbf{x},y)$.

$$\mathbf{E}_{(\mathbf{x},y)\sim p(\mathbf{x},y|t,s=-1)}[\ell(f(\mathbf{x},t),y)]$$

$$= \int \frac{\sum_z p(z|s=1)p(\mathbf{x},y|z,s=1)}{\sum_{z'} p(z'|s=1)p(\mathbf{x},y|z',s=1)}\frac{p(\mathbf{x}|t,s=1)}{p(\mathbf{x}|t,s=1)}p(\mathbf{x},y|t,s=-1)\ell(f(\mathbf{x},t),y)d\mathbf{x}dy \tag{3}$$

$$= \int \sum_z \left(\frac{p(z|s=1)p(\mathbf{x},y|z,s=1)}{\sum_{z'} p(z'|s=1)p(\mathbf{x},y|z',s=1)}\frac{p(\mathbf{x}|t,s=1)}{p(\mathbf{x}|t,s=1)}p(\mathbf{x}|t,s=-1)p(y|\mathbf{x},t)\ell(f(\mathbf{x},t),y)\right)d\mathbf{x}dy$$
$$\tag{4}$$

$$= \int \sum_z \left(p(z|s=1)p(\mathbf{x},y|z,s=1)\frac{p(\mathbf{x}|t,s=1)p(y|\mathbf{x},t)}{\sum_{z'} p(z'|s=1)p(\mathbf{x},y|z',s=1)}\frac{p(\mathbf{x}|t,s=-1)}{p(\mathbf{x}|t,s=1)}\right. \tag{5}$$
$$\left. \ell(f(\mathbf{x},t),y)\right)d\mathbf{x}dy$$

$$= \mathbf{E}_{(\mathbf{x},y)\sim\sum_z p(z|s=1)p(\mathbf{x},y|z,s=1)}\left[\frac{p(\mathbf{x},y|t,s=1)}{\sum_{z'} p(z'|s=1)p(\mathbf{x},y|z',s=1)}\frac{p(\mathbf{x}|t,s=-1)}{p(\mathbf{x}|t,s=1)}\ell(f(\mathbf{x},t),y)\right] \tag{6}$$

Equation 5 signifies that we can train a hypothesis for task $t$ by minimizing the expected loss over the distribution of all tasks weighted by $r_t(\mathbf{x},y)$. This amounts to minimizing the expected loss with respect to the target distribution $p(\mathbf{x},y|t,s=-1)$. The resampling weights of Equation 2 have an intuitive interpretation: The first fraction accounts for the difference in the joint distributions across tasks, and the second fraction accounts for the covariate shift within the target task.

Equation 5 leaves us with the problem of estimating the product of two density ratios $r_t(\mathbf{x},y) = \frac{p(\mathbf{x},y|t,s=1)}{\sum_z p(z|s=1)p(\mathbf{x},y|z,s=1)}\frac{p(\mathbf{x}|t,s=-1)}{p(\mathbf{x}|t,s=1)}$. One might be tempted to train four separate density estimators, one for each of the two numerators and the two denominators. However, obtaining estimators for potentially high-dimensional densities is unnecessarily difficult because ultimately only a scalar weight is required for each example.

## 3.1 Discriminative Density Ratio Models

In this section, we derive a discriminative model that directly estimates the two factors $r_t^1(\mathbf{x},y) = \frac{p(\mathbf{x},y|t,s=1)}{\sum_z p(z|s=1)p(\mathbf{x},y|z,s=1)}$ and $r_t^2(\mathbf{x}) = \frac{p(\mathbf{x}|t,s=-1)}{p(\mathbf{x}|t,s=1)}$ of the resampling weights $r_t(\mathbf{x},y) = r_t^1(\mathbf{x},y)r_t^2(\mathbf{x})$ without estimating the individual densities.

We reformulate the first density ratio $r_t^1(\mathbf{x},y) = \frac{p(\mathbf{x},y|t,s=1)}{\sum_z p(z|s=1)p(\mathbf{x},y|z,s=1)}$ in terms of a conditional model $p(t|\mathbf{x},y,s=1)$. This conditional has the following intuitive meaning: Given that an instance $(\mathbf{x},y)$ has been drawn at random from the pool distribution $\sum_z p(z|s=1)p(\mathbf{x},y|z,s=1)$

over all tasks (including target task $t$); the probability that $(\mathbf{x}, y)$ originates from $p(\mathbf{x}, y|t, s=1)$ is $p(t|\mathbf{x}, y, s=1)$. The following equations assume that the prior on the size of the target sample is greater than zero, $p(t|s=1) > 0$. In Equation 7 Bayes' rule is applied to the numerator and $z$ is summed out in the denominator. Equation 8 follows by dropping the normalization factor $p(t|s=1)$ and by canceling $p(\mathbf{x}, y|s=1)$.

$$r_t^1(\mathbf{x}, y) \quad = \quad \frac{p(\mathbf{x}, y|t, s=1)}{\sum_z p(z|s=1)p(\mathbf{x}, y|z, s=1)} \quad = \quad \frac{p(t|\mathbf{x}, y, s=1)p(\mathbf{x}, y|s=1)}{p(t|s=1)p(\mathbf{x}, y|s=1)} \tag{7}$$

$$\propto \quad p(t|\mathbf{x}, y, s=1) \tag{8}$$

The significance of Equation 8 is that it shows how the first factor of the resampling weights $r_t^1(\mathbf{x}, y)$ can be determined without knowledge of any of the task densities $p(\mathbf{x}, y|z, s=1)$. The right hand side of Equation 8 can be evaluated based on a model $p(t|\mathbf{x}, y, s=1)$ that discriminates labeled instances of the target task against labeled instances of the pool of examples for all non-target tasks.

Similar to the first density ratio, the second density ratio $r_t^2(\mathbf{x}) = \frac{p(\mathbf{x}|t, s=-1)}{p(\mathbf{x}|t, s=1)}$ can be expressed using a conditional model $p(s=1|\mathbf{x}, t)$. In Equation 9 Bayes' rule is applied twice. The two terms of $p(\mathbf{x}|t)$ cancel each other out, $p(s=1|t)/p(s=-1|t)$ is just a normalization factor, and since $p(s=-1|\mathbf{x}, t) = 1 - p(s=1|\mathbf{x}, t)$, Equation 10 follows.

$$r_t^2(\mathbf{x}) \quad = \quad \frac{p(\mathbf{x}|t, s=-1)}{p(\mathbf{x}|t, s=1)} \quad = \quad \frac{p(s=-1|\mathbf{x}, t)p(\mathbf{x}|t)}{p(s=-1|t)} \frac{p(s=1|t)}{p(s=1|\mathbf{x}, t)p(\mathbf{x}|t)} \tag{9}$$

$$\propto \quad \frac{1}{p(s=1|\mathbf{x}, t)} - 1 \tag{10}$$

The significance of the above derivations is that instead of the four potentially high-dimensional densities in $r_t(\mathbf{x}, y)$, only two conditional distributions with binary variables (Equations 8 and 10) need to be estimated. One can apply any probabilistic classifier to this estimation.

## 3.2 Estimation of Discriminative Density Ratios

For estimation of $r_t^1(\mathbf{x}, y)$ we model $p(t|\mathbf{x}, y, s=1)$ of Equation 8 with a logistic regression model

$$p(t|\mathbf{x}, y, s=1, \mathbf{u}_t) = \frac{1}{1 + \exp(-\mathbf{u}_t^\mathsf{T} \Phi(\mathbf{x}, y))}$$

over model parameters $\mathbf{u}_t$ using a problem-specific feature mapping $\Phi(\mathbf{x}, y)$. We define this mapping for binary labels, $\Phi(\mathbf{x}, y) = \begin{bmatrix} \delta(y, +1)\Phi(\mathbf{x}) \\ \delta(y, -1)\Phi(\mathbf{x}) \end{bmatrix}$, where $\delta$ is the Kronecker delta. In the absence of prior knowledge about the similarity of classes, input features $\mathbf{x}$ of examples with different class labels $y$ are mapped to disjoint subsets of the feature vector. With this feature mapping the models for positive and negative examples do not interact and can be trained independently. Any suitable mapping $\Phi(\mathbf{x})$ can be applied. In [8], $p(t|\mathbf{x}, y, s=1)$ is modeled for all tasks jointly in single optimization problem with a soft-max model. Empirically, we observe that a separate binary logistic regression model (as described above) for each task yields more accurate results with the drawback of a slightly increased overall training time.

**Optimization Problem 1** *For task $t$: over parameters $\mathbf{u}_t$, maximize*

$$\sum_{(\mathbf{x}, y) \in L_t} \log p(t|\mathbf{x}, y, s=1, \mathbf{u}_t) + \sum_{(\mathbf{x}, y) \in L \setminus L_t} \log(1 - p(t|\mathbf{x}, y, s=1, \mathbf{u}_t)) - \frac{\mathbf{u}_t^\mathsf{T} \mathbf{u}_t}{2\sigma_\mathbf{u}}.$$

The solution of Optimization Problem 1 is a MAP estimate of the logistic regression using a Gaussian prior on $\mathbf{u}_t$. The estimated vector $\mathbf{u}_t$ leads to the first part of the weighting factor $\hat{r}_t^1(\mathbf{x}, y) \propto p(t|\mathbf{x}, y, s=1, \mathbf{u}_t)$ according to Equation 8. $\hat{r}_t^1(\mathbf{x}, y)$ is normalized so that the weighted empirical distribution over the pool $L$ sums to one, $\frac{1}{|L|} \sum_{(\mathbf{x}, y) \in L} \hat{r}_t^1(\mathbf{x}, y) = 1$.

According to Equation 10 density ratio $r_t^2(\mathbf{x}) = \frac{p(\mathbf{x}|t, s=-1)}{p(\mathbf{x}|t, s=1)} \propto \frac{1}{p(s=1|\mathbf{x}, t)} - 1$ can be inferred from $p(s = 1|\mathbf{x}, t)$ which is the likelihood that a given $\mathbf{x}$ for task $t$ originates from the training

distribution, as opposed to from the test distribution. A model of $p(s=1|\mathbf{x}, t)$ can be obtained by discriminating a sample governed by $p(\mathbf{x}|t, s=1)$ against a sample governed by $p(\mathbf{x}|t, s=-1)$ using a probabilistic classifier. Unlabeled test data $T_t$ is governed by $p(\mathbf{x}|t, s=-1)$. The labeled pool $L$ over all training examples weighted by $r_t^1(\mathbf{x}, y)$ can serve as a sample governed by $p(\mathbf{x}|t, s=1)$; the labels $y$ can be ignored for this step. Empirically, we find that using the weighted pool $L$ instead of just $L_t$ (as used by [4]) achieves better results because the former sample is larger. We model $p(s=1|\mathbf{x}, \mathbf{v}_t)$ of Equation 10 with a regularized logistic regression on target variable $s$ with parameters $\mathbf{v}_t$ (Optimization Problem 2). Labeled examples $L$ are weighted by the estimated first factor $\hat{r}_t^1(\mathbf{x}, y)$ using the outcome of Optimization Problem 1.

**Optimization Problem 2** *For task $t$: over parameters $\mathbf{v}_t$, maximize*

$$\sum_{(\mathbf{x},y)\in L} \hat{r}_t^1(\mathbf{x}, y) \log p(s=1|\mathbf{x}, \mathbf{v}_t) + \sum_{\mathbf{x}\in T_t} \log p(s=-1|\mathbf{x}, \mathbf{v}_t) - \frac{\mathbf{v}_t^\mathsf{T}\mathbf{v}_t}{2\sigma_\mathbf{v}}.$$

With the result of Optimization Problem 2 the estimate for the second factor is $\hat{r}_t^2(\mathbf{x}) \propto \frac{1}{p(s=1|\mathbf{x}, \mathbf{v}_t)} - 1$, according to Equation 10. $\hat{r}_t^2(\mathbf{x})$ is normalized so that the final weighted empirical distribution over the pool sums to one, $\frac{1}{|L|}\sum_{(\mathbf{x},y)\in L} \hat{r}_t^1(\mathbf{x}, y)\hat{r}_t^2(\mathbf{x}) = 1$.

### 3.3 Weighted Empirical Loss and Target Model

The learning procedure first determines resampling weights $\hat{r}_t(\mathbf{x}, y) = \hat{r}_t^1(\mathbf{x}, y)\hat{r}_t^2(\mathbf{x})$ by solving Optimization Problems 1 and 2. These weights can now be used to reweight the labeled pool over all tasks and train the target model for task $t$. Using the weights we can evaluate the expected loss over the weighted training data as displayed in Equation 11. It is the regularized empirical counterpart of Equation 6.

$$\mathbf{E}_{(\mathbf{x},y)\sim L}\left[\hat{r}_t^1(\mathbf{x}, y)\hat{r}_t^2(\mathbf{x})\ell(f(\mathbf{x}, t), y)\right] + \frac{\mathbf{w}_t^\mathsf{T}\mathbf{w}_t}{2\sigma_\mathbf{w}^2} \qquad (11)$$

Optimization Problem 3 minimizes Equation 11, the weighted regularized loss over the training data using a standard Gaussian log-prior with variance $\sigma_\mathbf{w}^2$ on the parameters $\mathbf{w}_t$. Each example is weighted by the two discriminatively estimated density fractions from Equations 8 and 10 using the solution of Optimization Problems 1 and 2.

**Optimization Problem 3** *For task $t$: over parameters $\mathbf{w}_t$, minimize*

$$\frac{1}{|L|}\sum_{(\mathbf{x},y)\in L} \hat{r}_t^1(\mathbf{x}, y)\hat{r}_t^2(\mathbf{x})\ell(f(\mathbf{x}, \mathbf{w}_t), y) + \frac{\mathbf{w}_t^\mathsf{T}\mathbf{w}_t}{2\sigma_\mathbf{w}^2}.$$

In order to train target models for all tasks, instances of Optimization Problems 1 to 3 are solved for each task.

## 4 Targeted Advertising

We study the benefit of distribution matching and other reference methods on targeted advertising for four web portals. The portals play the role of tasks. We manually assign topic labels, out of a fixed set of 373 topics, to all web pages on all portals. For each user the topics of the surfed pages are tracked and the topic counts are stored in cookies of the user's web browser. The average number of surfed topics per user over all portals is 17. The feature vector $\mathbf{x}$ of a specific surfer is the normalized 373 dimensional vector of topic counts.

A small proportion of users is asked to fill out a web questionnaire that collects sociodemographic user profiles. About 25% of the questionnaires get completely filled out (accepted) and in 75% of the cases the user rejects to fill out the questionnaire. The accepted questionnaires constitute the training data $L$. The completion of the questionnaire cannot be enforced and it is therefore not possible to obtain labeled data that is governed by the test distribution of all users that surf the target portal. In order to evaluate the model, we approximate the distribution of users who reject the questionnaire

as follows. We take users who have answered the very first survey question (gender) but have then discontinued the survey as an approximation of the reject set. We add the correct proportion (25%) of users who have taken the survey, and thereby construct a sample that is governed by an approximation of the test distribution. Consequently, in our experiments we use the binary target label $y \in \{\text{male}, \text{female}\}$. Table 1 gives an overview of the data set.

Table 1: Portal statistics: number of accepted, partially rejected, and test examples (mix of all partial reject (=75%) and 25% accept); ratio of male users in training (accept) and test set.

| portal | # accept | # partial reject | # test | % male training | % male test |
|--------|----------|------------------|--------|-----------------|-------------|
| family | 8073 | 2035 | 2713 | 53.8% | 46.6% |
| TV channel | 8848 | 1192 | 1589 | 50.5% | 50.1% |
| news 1 | 3051 | 149 | 199 | 79.4% | 76.7% |
| news 2 | 2247 | 143 | 191 | 73.0% | 76.0% |

We compare distribution matching on labeled and unlabeled data (Optimization Problems 1 to 3) and distribution matching only on labeled data by setting $\hat{r}_t^2(\mathbf{x}) = 1$ in Optimization Problem 3 to the following reference models. The first baseline is a one-size-fits-all model that directly trains a logistic regression on $L$ (setting $\hat{r}_t^1(\mathbf{x}, y)\hat{r}_t^2(\mathbf{x}) = 1$ in Optimization Problem 3). The second baseline is a logistic regression trained only on $L_t$, the training examples of the target task. Training only on the reweighted target task data and correcting for marginal shift with respect to the unlabeled test data is the third baseline [4].

The last reference method is a hierarchical Bayesian model. Evgeniou and Pontil [6] describe a feature mapping for regularized regression models that corresponds to hierarchical Bayes with Gaussian prior on the regression parameters of the tasks. Training a logistic regression with their feature mapping over training examples from all tasks is equivalent to a joint MAP estimation of all model parameters and the mean of the Gaussian prior.

We evaluate the methods using all training examples from non-target tasks and different numbers of training examples of the target task. From all available accept examples of the target task we randomly select a certain number (0-1600) of training examples. From the remaining accept examples of the target task we randomly select an appropriate number and add them to all partial reject examples of the target task so that the evaluation set has the right proportions as described above. We repeat this process ten times and report the average accuracies of all methods.

We use a logistic loss as the target loss of distribution matching in Optimization Problem 3 and all reference methods. We compare kernelized variants of Optimization Problems 1 to 3 with RBF, polynomial, and linear kernels and find the linear kernel to achieve the best performance on our data set. All reported results are based on models with linear kernels. For the optimization of the logistic regression models we use trust region Newton descent [9].

We tune parameters $\sigma_{\mathbf{u}}$, $\sigma_{\mathbf{v}}$, and $\sigma_{\mathbf{w}}$ with grid search by executing the following steps.

1. $\sigma_{\mathbf{u}}$ is tuned by nested ten-fold cross-validation. The outer loop is a cross-validation on $L_t$. In each loop Optimization Problem 1 is solved on $L_{\neg t}$ merged with current training folds of $L_t$.

   - The inner loop temporarily tunes $\sigma_{\mathbf{w}}$ by cross-validation on rescaled $L_{\neg t}$ merged with the rescaled current training folds of $L_t$. At this point $\sigma_{\mathbf{w}}$ cannot be finally tuned because $\sigma_{\mathbf{v}}$ has not been tuned yet. In each loop Optimization Problem 3 is solved with fixed $\hat{r}_t^2(\mathbf{x}) = 1$. The temporary $\sigma_{\mathbf{w}}$ is chosen to maximize the accuracy on the tuning folds.

   Optimization Problem 3 is solved for each outer loop with the temporary $\sigma_{\mathbf{w}}$ and with $\hat{r}_t^2(\mathbf{x}) = 1$. The final $\sigma_{\mathbf{u}}$ is chosen to maximize the accuracy on the tuning folds of $L_t$ over all outer loops.

2. $\sigma_{\mathbf{v}}$ is tuned by likelihood cross-validation on $T_t \cup L$. The labels of the labeled data are ignored for this step. Test data $T_t$ of the target task as well as the weighted pool $L$ (weighted by $\hat{r}_t^1(\mathbf{x}, y)$, based on previously tuned $\sigma_{\mathbf{u}}$) are split into ten folds. With the nine training folds of the test data and the nine training folds of the weighted pool $L$, Optimization Problem 2 is solved. Parameter

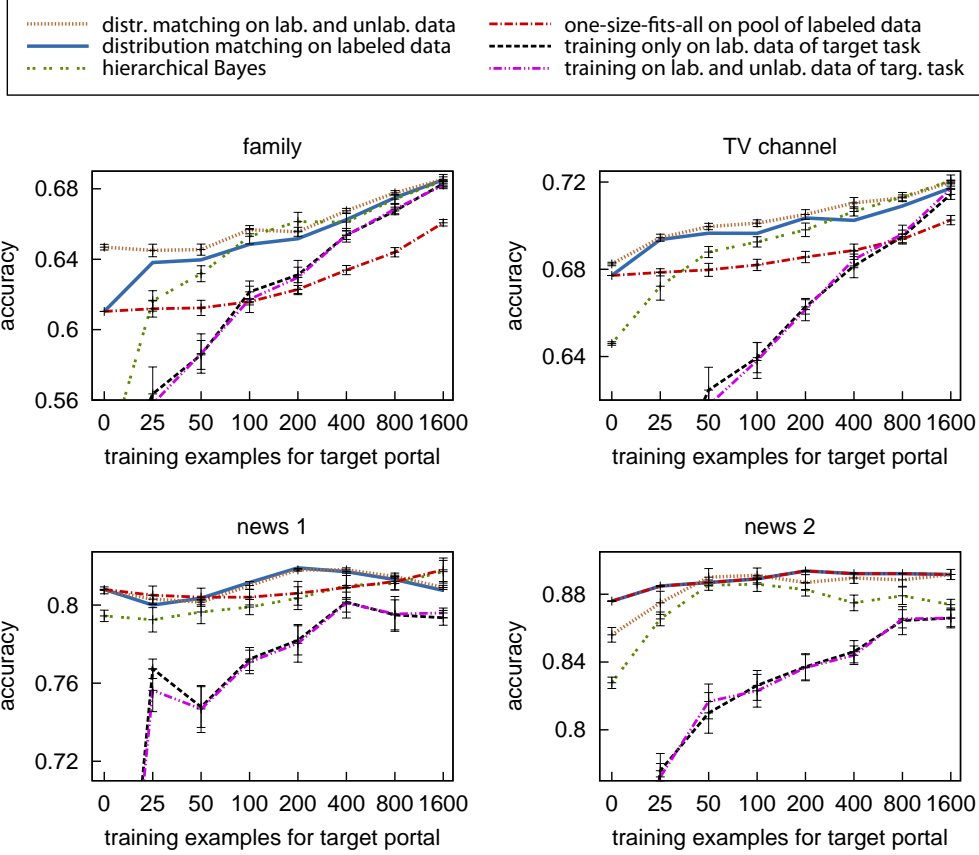

Figure 1: Accuracy over different number of training examples for target portal. Error bars indicate the standard error of the differences to distribution matching on labeled data.

$\sigma_{\mathbf{v}}$ is chosen to maximize the log-likelihood

$$\sum_{(\mathbf{x},y)\in L^{tune}} \hat{r}_t^1(\mathbf{x}, y) \log p(s{=}1|\mathbf{x}, \mathbf{v}_t) + \sum_{\mathbf{x}\in T_t^{tune}} \log p(s{=}{-}1|\mathbf{x}, \mathbf{v}_t)$$

on the tuning folds of test data and weighted pool (denoted by $L^{tune}$ and $T_t^{tune}$) over all ten cross-validation loops.

Applying non-uniform weights to labeled data (some of which may even be zero) reduces the effective sample size. This leads to a bias-variance trade-off [1]: training on unweighted data causes a bias, applying non-uniform weights reduces the sample size and increases the variance of the estimator. We follow [1] and smooth the estimated weights by $\hat{r}_t^2(\mathbf{x})^\lambda$ before including them into Optimization Problem 3. The smoothing parameter $\lambda$ biases the weights towards uniformity and thereby controls the trade-off. Without looking at the test data of the target task we tune $\eta$ on the non-target tasks so that the accuracy of the distribution matching method is maximized. This procedure usually results in $\eta$ values around 0.3.

3. Finally, $\sigma_{\mathbf{w}}$ is tuned by cross-validation on $L$ rescaled by $\hat{r}_t^1(\mathbf{x}, y)\hat{r}_t^2(\mathbf{x})$ (based on the previously tuned parameters $\sigma_{\mathbf{u}}$ and $\sigma_{\mathbf{v}}$). In each cross-validation loop Optimization Problem 3 is solved.

Figure 1 displays the accuracies over different numbers of labeled data for the four different target portals. The error bars are the standard errors of the differences to the distribution matching method on labeled data (solid blue line).

For the "family" and "TV channel" portals the distribution matching method on labeled and unlabeled data outperforms all other methods in almost all cases. The distribution matching method on

labeled data outperforms the baselines trained only on the data of the target task for all portals and all data set sizes and it is at least as good as the one-size-fits-all model in almost all cases. The hierarchical Bayesian method yields low accuracies for smaller numbers of training examples but becomes comparable to the distribution matching method when training set sizes of the target portal increase. The simple covariate shift model that trains only on labeled and unlabeled data of the target task does not improve over the *iid* model that only trains on the labeled data of the target task. This indicates that the marginal shift between training and test distributions is small, or could indicate that the approximation of the reject distribution which we use in our experimentation is not sufficiently close. Either reason also explains why accounting for the marginal shift in the distribution matching method does not always improve over distribution matching using only labeled data.

Transfer learning by distribution matching passes all examples for all tasks to the underlying logistic regressions. This is computationally more expensive than the reference methods. For example, the single task baseline trains only one logistic regression on the examples of the target task. Empirically, we observe that all methods scale linearly in the number training examples.

## 5  Conclusion

We derived a multi-task learning method that is based on the insight that the expected loss with respect to the unbiased test distribution of the target task is equivalent to the expected loss over the biased training examples of all tasks weighted by a task specific resampling weight. This led to an algorithm that discriminatively estimates these resampling weights by training two simple conditional models. After weighting the pooled examples over all tasks the target model for a specific task can be trained.

In our empirical study on targeted advertising, we found that distribution matching using labeled data outperforms all reference methods in almost all cases; the differences are particularly large for small sample sizes. Distribution matching with labeled and unlabeled data outperforms the reference methods and distribution matching with only labeled data in two out of four portals. Even with no labeled data of the target task the performance of the distribution matching method is comparable to training on 1600 examples of the target task for all portals.

### Acknowledgments

We gratefully acknowledge support by nugg.ad AG and the German Science Foundation DFG. We wish to thank Stephan Noller and the nugg.ad team for their valuable contributions.

## References

[1] H. Shimodaira. Improving predictive inference under covariate shift by weighting the log-likelihood function. *Journal of Statistical Planning and Inference*, 90:227–244, 2000.

[2] J. Huang, A. Smola, A. Gretton, K. Borgwardt, and B. Schölkopf. Correcting sample selection bias by unlabeled data. In *Advances in Neural Information Processing Systems*, 2007.

[3] M. Sugiyama, S. Nakajima, H. Kashima, P. von Bunau, and M. Kawanabe. Direct importance estimation with model selection and its application to covariate shift adaptation. In *Advances in Neural Information Processing Systems*, 2008.

[4] S. Bickel, M. Brückner, and T. Scheffer. Discriminative learning for differing training and test distributions. In *Proceedings of the International Conference on Machine Learning*, 2007.

[5] A. Schwaighofer, V. Tresp, and K. Yu. Learning Gaussian process kernels via hierarchical Bayes. In *Advances in Neural Information Processing Systems*, 2005.

[6] T. Evgeniou and M. Pontil. Regularized multi–task learning. *Proceedings of the International Conference on Knowledge Discovery and Data Mining*, pages 109–117, 2004.

[7] Y. Xue, X. Liao, L. Carin, and B. Krishnapuram. Multi-task learning for classification with Dirichlet process priors. *Journal of Machine Learning Research*, 8:35–63, 2007.

[8] S. Bickel, J. Bogojeska, T. Lengauer, and T. Scheffer. Multi-task learning for HIV therapy screening. In *Proceedings of the International Conference on Machine Learning*, 2008.

[9] C. Lin, R. Weng, and S. Keerthi. Trust region Newton method for large-scale logistic regression. *Journal of Machine Learning Research*, 9:627–650, 2008.
